# Sparse Manifold Clustering and Embedding

**Ehsan Elhamifar**
Center for Imaging Science
Johns Hopkins University
ehsan@cis.jhu.edu

**René Vidal**
Center for Imaging Science
Johns Hopkins University
rvidal@cis.jhu.edu

## Abstract

We propose an algorithm called Sparse Manifold Clustering and Embedding (SMCE) for simultaneous clustering and dimensionality reduction of data lying in multiple nonlinear manifolds. Similar to most dimensionality reduction methods, SMCE finds a small neighborhood around each data point and connects each point to its neighbors with appropriate weights. The key difference is that SMCE finds both the neighbors and the weights automatically. This is done by solving a sparse optimization problem, which encourages selecting nearby points that lie in the same manifold and approximately span a low-dimensional affine subspace. The optimal solution encodes information that can be used for clustering and dimensionality reduction using spectral clustering and embedding. Moreover, the size of the optimal neighborhood of a data point, which can be different for different points, provides an estimate of the dimension of the manifold to which the point belongs. Experiments demonstrate that our method can effectively handle multiple manifolds that are very close to each other, manifolds with non-uniform sampling and holes, as well as estimate the intrinsic dimensions of the manifolds.

## 1 Introduction

### 1.1 Manifold Embedding

In many areas of machine learning, pattern recognition, information retrieval and computer vision, we are confronted with high-dimensional data that lie in or close to a manifold of intrinsically low-dimension. In this case, it is important to perform dimensionality reduction, *i.e.*, to find a compact representation of the data that unravels their few degrees of freedom.

The first step of most dimensionality reduction methods is to build a neighborhood graph by connecting each data point to a fixed number of nearest neighbors or to all points within a certain radius of the given point. Local methods, such as LLE [1], Hessian LLE [2] and Laplacian eigenmaps (LEM) [3], try to preserve local relationships among points by learning a set of weights between each point and its neighbors. Global methods, such as Isomap [4], Semidefinite embedding [5], Minimum volume embedding [6] and Structure preserving embedding [7], try to preserve local and global relationships among all data points. Both categories of methods find the low-dimensional representation of the data from a few eigenvectors of a matrix related to the learned weights between pairs of points.

For both local and global methods, a proper choice of the neighborhood size used to build the neighborhood graph is critical. Specifically, a small neighborhood size may not capture sufficient information about the manifold geometry, especially when it is smaller than the intrinsic dimension of the manifold. On the other hand, a large neighborhood size could violate the principles used to capture information about the manifold. Moreover, the curvature of the manifold and the density of the data points may be different in different regions of the manifold, hence using a fix neighborhood size may be inappropriate.

## 1.2 Manifold Clustering

In many real-world problems, the data lie in multiple manifolds of possibly different dimensions. Thus, to find a low-dimensional embedding of the data, one needs to first cluster the data according to the underlying manifolds and then find a low-dimensional representation for the data in each cluster. Since the manifolds can be very close to each other and they can have arbitrary dimensions, curvature and sampling, the manifold clustering and embedding problem is very challenging.

The particular case of clustering data lying in multiple flat manifolds (subspaces) is well studied and numerous algorithms have been proposed (see *e.g.*, the tutorial [8]). However, such algorithms take advantage of the global linear relations among data points in the same subspace, hence they cannot handle nonlinear manifolds. Other methods assume that the manifolds have different instrinsic dimensions and cluster the data according to the dimensions rather than the manifolds themselves [9, 10, 11, 12, 13]. However, in many real-world problems this assumption is violated. Moreover, estimating the dimension of a manifold from a point cloud is a very difficult problem on its own.

When manifolds are densely sampled and sufficiently separated, existing dimensionality reduction algorithms such as LLE can be extended to perform clustering before the dimensionality reduction step [14, 15, 16]. More precisely, if the size of the neighborhood used to build the similarity graph is chosen to be small enough not to include points from other manifolds and large enough to capture the local geometry of the manifold, then the similarity graph will have multiple connected components, one per manifold. Therefore, spectral clustering methods can be employed to separate the data according to the connected components. However, as we will see later, finding the right neighborhood size is in general difficult, especially when manifolds are close to each other. Moreover, in some cases one cannot find a neighborhood that contains only points from the same manifold.

## 1.3 Paper Contributions

In this paper, we propose an algorithm, called SMCE, for *simultaneous clustering and embedding of data lying in multiple manifolds*. To do so, we use the geometrically motivated assumption that for each data point there exists a small neighborhood in which only the points that come from the same manifold lie approximately in a low-dimensional affine subspace. We propose an optimization program based on sparse representation to select *a few neighbors* of each data point that span a *low-dimensional affine subspace* passing near that point. As a result, a few nonzero elements of the solution indicate the points that are on the same manifold, hence they can be used for clustering. In addition, the weights associated to the chosen neighbors indicate their distances to the given data point, which can be used for dimensionality reduction. Thus, unlike conventional methods that first build a neighborhood graph and then extract information from it, our method simultaneously builds the neighborhood graph and obtains its weights. This leads to successful results even in challenging situations where the nearest neighbors of a point come from other manifolds. Clustering and embedding of the data into lower dimensions follows by taking the eigenvectors of the matrix of weights and its submatrices, which are sparse hence can be stored and be operated on efficiently. Thanks to the sparse representations obtained by SMCE, the number of neighbors of the data points in each manifold reflects the intrinsic dimensionality of the underlying manifold. Finally, SMCE has only one free parameter that, for a large range of variation, results in a stable clustering and embedding, as the experiments will show. To the best of our knowledge, SMCE is the only algorithm proposed to date that allows robust automatic selection of neighbors and simultaneous clustering and dimensionality reduction in a *unified manner*.

## 2 Proposed Method

Assume we are given a collection of $N$ data points $\{\boldsymbol{x}_i \in \mathbb{R}^D\}_{i=1}^N$ lying in $n$ different manifolds $\{\mathcal{M}_l\}_{l=1}^n$ of intrinsic dimensions $\{d_l\}_{l=1}^n$. In this section, we consider the problem of simultaneously clustering the data according to the underlying manifolds and obtaining a low-dimensional representation of the data points within each cluster.

We approach this problem using a spectral clustering and embedding algorithm. Specifically, we build a similarity graph whose nodes represent the data points and whose edges represent the similarity between data points. The fundamental challenge is to decide which nodes should be connected and how. To do clustering, we wish to connect each point to other points from the same manifold. To

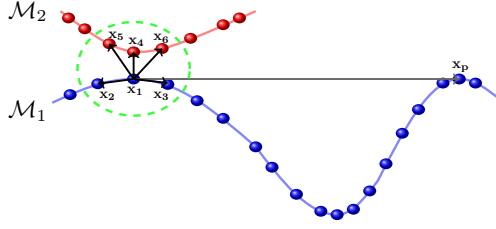

Figure 1: For $\boldsymbol{x}_1 \in \mathcal{M}_1$, the smallest neighborhood containing points from $\mathcal{M}_1$ also contains points from $\mathcal{M}_2$. However, only the neighbors in $\mathcal{M}_1$ span a 1-dimensional subspace around $\boldsymbol{x}_1$.

do dimensionality reduction, we wish to connect each point to neighboring points with appropriate weights that reflect the neighborhood information. To simultaneously pursue both goals, we wish to select neighboring points from the same manifold.

We address this problem by formulating an optimization algorithm based on sparse representation. The underlying assumption behind the proposed method is that each data point has a small neighborhood in which the minimum number of points that span a low-dimensional affine subspace passing near that point is given by the points from the same manifold. More precisely:

**Assumption 1** *For each data point $\boldsymbol{x}_i \in \mathcal{M}_l$ consider the smallest ball $\mathcal{B}_i \subset \mathbb{R}^D$ that contains the $d_l + 1$ nearest neighbors of $\boldsymbol{x}_i$ from $\mathcal{M}_l$. Let the neighborhood $\mathcal{N}_i$ be the set of all data points in $\mathcal{B}_i$ excluding $\boldsymbol{x}_i$. In general, this neighborhood contains points from $\mathcal{M}_l$ as well as other manifolds. We assume that for all $i$ there exists $\epsilon \geq 0$ such that the nonzero entries of the sparsest solution of*

$$\| \sum_{j \in \mathcal{N}_i} c_{ij}(\boldsymbol{x}_j - \boldsymbol{x}_i) \|_2 \leq \epsilon \ \ and \ \ \sum_{j \in \mathcal{N}_i} c_{ij} = 1 \tag{1}$$

*corresponds to the $d_l + 1$ neighbors of $\boldsymbol{x}_i$ from $\mathcal{M}_l$. In other words, among all affine subspaces spanned by subsets of the points $\{\boldsymbol{x}_j\}_{j \in \mathcal{N}_i}$ and passing near $\boldsymbol{x}_i$ up to $\epsilon$ error, the one of lowest dimension has dimension $d_l$ and it is spanned by the $d_l + 1$ neighbors of $\boldsymbol{x}_i$ from $\mathcal{M}_l$.*

In the limiting case of densely sampled data, this affine subspace coincides with the $d_l$-dimensional tangent space of $\mathcal{M}_l$ at $\boldsymbol{x}_i$. To illustrate this, consider the two manifolds shown in Figure 1 and assume that points $\boldsymbol{x}_4$, $\boldsymbol{x}_5$ and $\boldsymbol{x}_6$ are closer to $\boldsymbol{x}_1$ than $\boldsymbol{x}_2$ or $\boldsymbol{x}_3$. Then any small ball centered at $\boldsymbol{x}_1 \in \mathcal{M}_1$ that contains $\boldsymbol{x}_2$ and $\boldsymbol{x}_3$ will also contain points $\boldsymbol{x}_4$, $\boldsymbol{x}_5$ and $\boldsymbol{x}_6$. In this case, among affine spans of all possible choices of 2 points in this neighborhood, the one corresponding to $\boldsymbol{x}_2$ and $\boldsymbol{x}_3$ is the closest one to $\boldsymbol{x}_1$, and is also close to the tangent space of $\mathcal{M}_1$ at $\boldsymbol{x}_1$. On the other hand, the affine span of any choices of 3 or more data points in the neighborhood always passes through $\boldsymbol{x}_1$. However, this requires a linear combination of more than 2 data points.

### 2.1 Optimization Algorithm

Our goal is to propose a method that selects, for each data point $\boldsymbol{x}_i$, a few neighbors that lie in the same manifold. If the neighborhood $\mathcal{N}_i$ is known and of relatively small size, one can search for the minimum number of points that satisfy (1). However, $\mathcal{N}_i$ is not known a priori and searching for a few data points in $\mathcal{N}_i$ that satisfy (1) becomes more computationally complex as the size of the neighborhood increases. To tackle this problem, we let the size of the neighborhood be arbitrary. However, by using a sparse optimization program, we bias the method to select a few data points that are close to $\boldsymbol{x}_i$ and span a low-dimensional affine subspace passing near $\boldsymbol{x}_i$.

Consider a point $\boldsymbol{x}_i$ in the $d_l$-dimensional manifold $\mathcal{M}_l$ and consider the set of points $\{\boldsymbol{x}_j\}_{j \neq i}$. It follows from Assumption 1 that, among these points, the ones that are neighbors of $\boldsymbol{x}_i$ in $\mathcal{M}_l$ span a $d_l$-dimensional affine subspace of $\mathbb{R}^D$ that passes near $\boldsymbol{x}_i$. In other words,

$$\| [\boldsymbol{x}_1 - \boldsymbol{x}_i \ \cdots \ \boldsymbol{x}_N - \boldsymbol{x}_i] \, \boldsymbol{c}_i \|_2 \leq \epsilon \ \ and \ \ \mathbf{1}^\top \boldsymbol{c}_i = 1 \tag{2}$$

has a solution $\boldsymbol{c}_i$ whose $d_l + 1$ nonzero entries corresponds to $d_l + 1$ neighbors of $\boldsymbol{x}_i$ in $\mathcal{M}_l$.

Notice that after relaxing the size of the neighborhood, the solution $\boldsymbol{c}_i$ that uses the minimum number of data points, *i.e.*, the solution $\boldsymbol{c}_i$ with the smallest number of nonzero entries, may no longer be

unique. In the example of Figure 1, for instance, a solution of (2) with two nonzero entries can correspond to an affine combination of $\boldsymbol{x}_2$ and $\boldsymbol{x}_3$ or an affine combination of $\boldsymbol{x}_2$ and $\boldsymbol{x}_p$. To bias the solutions of (2) to the one that corresponds to the closest neighbors of $\boldsymbol{x}_i$ in $\mathcal{M}_l$, we set up an optimization program whose objective function favors selecting a few neighbors of $\boldsymbol{x}_i$ subject to the constraint in (2), which enforces selecting points that approximately lie in an affine subspace at $\boldsymbol{x}_i$. Before that, it is important to decouple the goal of selecting a few neighbors from that of spanning an affine subspace. To do so, we normalize the vectors $\{\boldsymbol{x}_j - \boldsymbol{x}_i\}_{j \neq i}$ and let

$$\boldsymbol{X}_i \triangleq \begin{bmatrix} \frac{\boldsymbol{x}_1 - \boldsymbol{x}_i}{\|\boldsymbol{x}_1 - \boldsymbol{x}_i\|_2} & \cdots & \frac{\boldsymbol{x}_N - \boldsymbol{x}_i}{\|\boldsymbol{x}_N - \boldsymbol{x}_i\|_2} \end{bmatrix} \in \mathbb{R}^{D \times N-1}. \tag{3}$$

In this way, for a small $\varepsilon$, the locations of the nonzero entries of any solution $\boldsymbol{c}_i$ of $\|\boldsymbol{X}_i \boldsymbol{c}_i\|_2 \leq \varepsilon$ do not depend on whether the selected points are close to or far from $\boldsymbol{x}_i$. Now, among all the solutions of $\|\boldsymbol{X}_i \boldsymbol{c}_i\|_2 \leq \varepsilon$ that satisfy $\mathbf{1}^\top \boldsymbol{c}_i = 1$, we look for the one that uses *a few closest neighbors* of $\boldsymbol{x}_i$. To that end, we consider an objective function that penalizes points based on their proximity to $\boldsymbol{x}_i$. That is, points that are closer to $\boldsymbol{x}_i$ get lower penalty than points that are farther away. We thus consider the following weighted $\ell_1$-optimization program

$$\min \|\boldsymbol{Q}_i \boldsymbol{c}_i\|_1 \quad \text{subject to} \quad \|\boldsymbol{X}_i \boldsymbol{c}_i\|_2 \leq \varepsilon, \quad \mathbf{1}^\top \boldsymbol{c}_i = 1, \tag{4}$$

where the $\ell_1$-norm promotes sparsity of the solution [17] and the *proximity inducing* matrix $\boldsymbol{Q}_i$, which is a positive-definite diagonal matrix, favors selecting points that are close to $\boldsymbol{x}_i$. Note that the elements of $\boldsymbol{Q}_i$ should be chosen such that the points that are closer to $\boldsymbol{x}_i$ have smaller weights, allowing the assignment of nonzero coefficients to them. Conversely, the points that are farther from $\boldsymbol{x}_i$ should have larger weights, favoring the assignment of zero coefficients to them. A simple choice of the proximity inducing matrix is to select the diagonal elements of $\boldsymbol{Q}_i$ to be $\frac{\|\boldsymbol{x}_j - \boldsymbol{x}_i\|_2}{\sum_{t \neq i} \|\boldsymbol{x}_t - \boldsymbol{x}_i\|_2} \in (0, 1]$. Also, one can use other types of weights, such as exponential weights $\frac{\exp(\|\boldsymbol{x}_j - \boldsymbol{x}_i\|_2/\sigma)}{\sum_{t \neq i} \exp(\|\boldsymbol{x}_t - \boldsymbol{x}_i\|_2/\sigma)}$ where $\sigma > 0$. However, the former choice of the weights, which is also tuning parameter free, works very well in practice, as we will show later.

Another optimization program which is related to (4) by the method of Lagrange multipliers, is

$$\min \lambda \|\boldsymbol{Q}_i \boldsymbol{c}_i\|_1 + \frac{1}{2} \|\boldsymbol{X}_i \boldsymbol{c}_i\|_2^2 \quad \text{subject to} \quad \mathbf{1}^\top \boldsymbol{c}_i = 1, \tag{5}$$

where the parameter $\lambda$ sets the trade-off between the sparsity of the solution and the affine reconstruction error. Notice that this new optimization program, which also prefers sparse solutions, is similar to the Lasso optimization problem [18, 17]. The only modification, is the introduction of the affine constraint $\mathbf{1}^\top \boldsymbol{c}_i = 1$. As we will show in the next section, there is a wide range of values of $\lambda$ for which the optimization program in (5) successfully finds a sparse solution for each point from neighbors in the same manifold.

Notice that, in sharp contrast to the nearest neighbors-based methods, which first fix the number of neighbors or the neighborhood radius and then compute the weights between points in each neighborhood, we do the two steps at the same time. In other words, the optimization programs (4) and (5) *automatically* choose a few neighbors of the given data point, which approximately span a low-dimensional affine subspace at that point. In addition, by the definition of $\boldsymbol{Q}_i$ and $\boldsymbol{X}_i$, the solutions of the optimization programs (4) and (5) are invariant with respect to a global rotation, translation, and scaling of the data points.

## 2.2 Clustering and Dimensionality Reduction

By solving the proposed optimization programs for each data point, we obtain the necessary information for clustering and dimensionality reduction. This is because the solution $\boldsymbol{c}_i^\top \triangleq [c_{i1} \cdots c_{iN}]$ of the proposed optimization programs satisfies

$$\sum_{j \neq i} \frac{c_{ij}}{\|\boldsymbol{x}_j - \boldsymbol{x}_i\|_2} (\boldsymbol{x}_j - \boldsymbol{x}_i) \approx \boldsymbol{0}. \tag{6}$$

Hence, we can rewrite $\boldsymbol{x}_i \approx \begin{bmatrix} \boldsymbol{x}_1 & \boldsymbol{x}_2 & \cdots & \boldsymbol{x}_N \end{bmatrix} \boldsymbol{w}_i$, where the weight vector $\boldsymbol{w}_i^\top \triangleq [w_{i1} \cdots w_{iN}] \in \mathbb{R}^N$ associated to the $i$-th data point is defined as

$$w_{ii} \triangleq 0, \quad w_{ij} \triangleq \frac{c_{ij}/\|\boldsymbol{x}_j - \boldsymbol{x}_i\|_2}{\sum_{t \neq i} c_{it}/\|\boldsymbol{x}_t - \boldsymbol{x}_i\|_2}, \quad j \neq i. \tag{7}$$

The indices of the few nonzero elements of $\boldsymbol{w}_i$, ideally, correspond to neighbors of $\boldsymbol{x}_i$ in the same manifold and their values indicate their (inverse) distances to $\boldsymbol{x}_i$.

Next, we use the weights $\boldsymbol{w}_i$ to perform clustering and dimensionality reduction. We do so by building a similarity graph $\mathcal{G} = (V, E)$ whose nodes represent the data points. We connect each node $i$, corresponding to $\boldsymbol{x}_i$, to the node $j$, corresponding to $\boldsymbol{x}_j$, with an edge whose weight is equal to $|w_{ij}|$. While, potentially, every node can get connected to all other nodes, because of the sparsity of $\boldsymbol{w}_i$, each node $i$ connects itself to only a few other nodes that correspond to the neighbors of $\boldsymbol{x}_i$ in the same manifold. We call such neighbors as *sparse neighbors*. In addition, the distances of the sparse neighbors to $\boldsymbol{x}_i$ are reflected in the weights $|w_{ij}|$.

The similarity graph built in this way has ideally several connected components, where points in the same manifold are connected to each other and there is no connection between two points in different manifolds. In other words, the similarity matrix of the graph has ideally the following form

$$\boldsymbol{W} \triangleq [\, |\boldsymbol{w}_1| \;\; \cdots \;\; |\boldsymbol{w}_N| \,] = \begin{bmatrix} \boldsymbol{W}[1] & \boldsymbol{0} & \cdots & \boldsymbol{0} \\ \boldsymbol{0} & \boldsymbol{W}[2] & \cdots & \boldsymbol{0} \\ \vdots & \vdots & \ddots & \vdots \\ \boldsymbol{0} & \boldsymbol{0} & \cdots & \boldsymbol{W}[n] \end{bmatrix} \boldsymbol{\Gamma}, \tag{8}$$

where $\boldsymbol{W}[l]$ is the similarity matrix of the data points in $\mathcal{M}_l$ and $\boldsymbol{\Gamma} \in \mathbb{R}^{N \times N}$ is an unknown permutation matrix. Clustering of the data follows by applying spectral clustering [19] to $\boldsymbol{W}$.[1] One can also determine the number of connected components by analyzing the eigenspectrum of the Laplacian matrix [20].

Any of the existing dimensionality reduction techniques can be applied to the data in each cluster to obtain a low-dimensional representation of the data in the corresponding manifold. However, this would require new computation of neighborhoods and weights. On the other hand, the similarity graph built by our method has a locality preserving property by the definition of the weights. Thus, we can use the adjacency matrix, $\boldsymbol{W}[i]$, of the $i$-th cluster as a similarity between points in the corresponding manifold and obtain a low-dimensional embedding of the data by taking the last few eigenvectors of the normalized Laplacian matrix associated to $\boldsymbol{W}[i]$ [3]. Note that there are other ways for inferring the low-dimensional embedding of the data in each cluster along the line of [21] and [1] which is beyond the scope of the current paper.

## 2.3 Intrinsic Dimension Information

An advantage of proposed sparse optimization algorithm is that it provides information about the intrinsic dimension of the manifolds. This comes from the fact that a data point $\boldsymbol{x}_i \in \mathcal{M}_l$ and its neighbors in $\mathcal{M}_l$ lie approximately in the $d_l$-dimensional tangent space of $\mathcal{M}_l$ at $\boldsymbol{x}_i$. Since $d_l + 1$ vectors in this tangent space are linearly dependent, the solution $\boldsymbol{c}_i$ of the proposed optimization programs is expected to have $d_l + 1$ nonzero elements. As a result, we can obtain information about the intrinsic dimension of the manifolds in the following way. Let $\Omega_l$ denote the set of indices of points that belong to the $l$-th cluster. For each point in $\Omega_l$, we sort the elements of $|\boldsymbol{c}_i|$ from the largest to the smallest and denote the new vector as $\boldsymbol{c}_{s,i}$. We define the *median sparse coefficient* vector of the $l$-th cluster as

$$\boldsymbol{msc}^{(l)} = \text{median}\{\boldsymbol{c}_{s,i}\}_{i \in \Omega_l}, \tag{9}$$

whose $j$-th element is computed as the median of the $j$-th elements of the vectors $\{\boldsymbol{c}_{s,i}\}_{i \in \Omega_l}$. Thus, the number of nonzero elements of $\boldsymbol{msc}^{(l)}$ or, more practically, the number of elements with relatively high magnitude, gives an estimate of the intrinsic dimension of the $l$-th manifold plus one.[2]

An advantage of our method is that it allows us to have a different neighborhood size for each data point, depending on the local dimension of its underlying manifold at that point. For example, in the case of two manifolds of dimensions $d_1 = 2$ and $d_2 = 30$, for data points in the $l$-th manifold we automatically obtain solutions with $d_l + 1$ nonzero elements. On the other hand, methods that fix the number of neighbors fall into trouble because the number of neighbors would be too small for one manifold or too large for the other manifold.

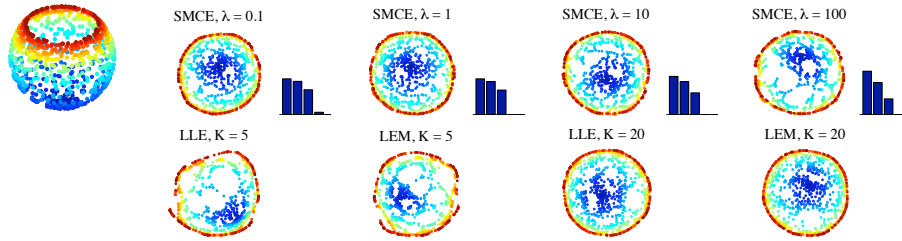

Figure 2: Top: embedding of a punctured sphere and the $msc$ vectors obtained by SMCE for different values of $\lambda$. Bottomn: embedding obtained by LLE and LEM for different values of $K$.

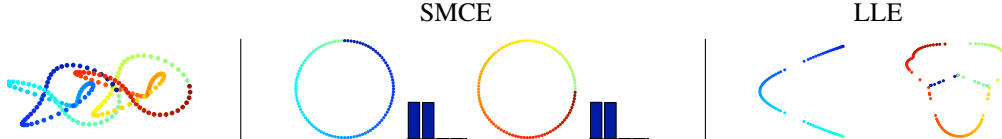

Figure 3: Clustering and embedding for two trefoil-knots. Left: original manifolds. Middle: embedding and $msc$ vectors obtained by SMCE. Right: clustering and embedding obtained by LLE.

## 3   Experiments

In this section, we evaluate the performance of SMCE on a number of synthetic and real experiments. For all the experiments, we use the optimization program (5), where we typically set $\lambda = 10$. However, the clustering and embedding results obtained by SMCE are stable for $\lambda \in [1, 200]$. Since the weighted $\ell_1$-optimization does not select the points that are very far from the given point, we consider only $L < N - 1$ neighbors of each data point in the optimization program, where we typically set $L = N/10$. As in the case of nearest neighbors-based methods, there is no guarantee that the points in the same manifold form a single connected component of the similarity graph built by SMCE. However, this has always been the case in our experiments, as we will show next.

### 3.1   Experiments with Synthetic Data

**Manifold Embedding.**   We first evaluate SMCE for the dimensionality reduction task only. We sample $N = 1,000$ data points from a 2-sphere, where a neighborhood of its north pole is excluded. We then embed the data in $\mathbb{R}^{100}$, add small Gaussian white noise to it and apply SMCE for $\lambda \in \{0.1, 1, 10, 100\}$. Figure 2 shows the embedding results of SMCE in a 2 dimensional Euclidean space. The three large elements of the $msc$ vector for different values of $\lambda$ correctly reflect the fact that the sphere has dimension two. However, note that for very large values of $\lambda$ the performance of the embedding degrades since we put more emphasis on the sparsity of the solution. The results in the bottom of Figure 2 show the embeddings obtained by LLE and LEM for $K = 5$ and $K = 20$ nearest neighbors. Notice that, for $K = 20$, nearest neighbor-based methods obtain similar embedding results to those of SMCE, while for $K = 5$ they obtain poor embedding results. This suggests that the principle used by SMCE to select the neighbors is very effective: it chooses very few neighbors that are very informative for dimensionality reduction.

**Manifold Clustering and Embedding.**   Next, we consider the challenging case where the manifolds are close to each other. We consider two trefoil-knots, shown in Figure 3, which are embedded in $\mathbb{R}^{100}$ and are corrupted with small Gaussian white noise. The data points are sampled such that among the 2 nearest neighbors of 1% of the data points there are points from the other manifold. Also, among the 3 and 5 nearest neighbors of 9% and 18% of the data points, respectively, there are points from the other manifold. For such points, the nearest neighbors-based methods will connect them to nearby points in the other manifold and assign large weights to the connection. As a result, these methods cannot obtain a proper clustering or a successful embedding. Table 1 shows the misclassification rates of LLE and LEM for different number of nearest neighbors $K$ as well as the misclassification rates of SMCE for different values of $\lambda$. While there is no $K$ for which we can successfully cluster the data using LLE and LEM, for a wide range of $\lambda$, SMCE obtains a perfect clustering. Figure 3 shows the results of SMCE for $\lambda = 10$ and LLE for $K = 3$. As the results

Table 1: Misclassifications rates for LLE and LEM as a function of $K$ and for SMCE as a function of $\lambda$.

| $K$ | 2 | 3 | 4 | 5 | 6 | 8 | 10 |
|---|---|---|---|---|---|---|---|
| LLE | 15.5% | 9.5% | 16.5% | 13.5% | 16.5% | 37.5 | 38.5% |
| LEM | 15.5% | 13.5% | 17.5% | 14.5% | 28.5% | 28.5% | 13.5% |

| $\lambda$ | 0.1 | 1 | 10 | 50 | 70 | 100 | 200 |
|---|---|---|---|---|---|---|---|
| SMCE | 15.5% | 6.0% | 0.0% | 0.0% | 0.0% | 0.0% | 0.0% |

Table 2: Percentage of data points whose $K$ nearest neighbors contain points from the other manifold.

| $K$ | 1 | 2 | 3 | 4 | 7 | 10 |
|---|---|---|---|---|---|---|
| | 3.9% | 10.2% | 23.4% | 35.2% | 57.0% | 64.8% |

show, enforcing that the neighbors of a point from the same manifold span a low-dimensional affine subspace helps to select neighbors from the correct manifold and not from the other manifolds. This results in successful clustering and embedding of the data as well as unraveling the dimensions of the manifolds. On the other hand, the fact that LLE and LEM choose wrong neighbors, results in a low quality embedding.

## 3.2 Experiments with Real Data

In this section, we examine the performance of SMCE on real datasets. We show that challenges such as manifold proximity and non-uniform sampling are also common in real data sets, and that our algorithm is able to handle these issues effectively.

First, we consider the problem of clustering and embedding of face images of two different subjects from the Extended Yale B database [22]. Each subject has 64 images of $192 \times 168$ pixels captured under a fixed pose and expression and with varying illuminations. By applying SMCE with $\lambda = 10$ on almost $33,000$-dimensional vectorized faces, we obtain a misclassification rate of $2.34\%$, which corresponds to wrongly clustering 3 out of the 128 data points. Figure 4, top row, shows the embeddings obtained by SMCE, LLE and LEM for the whole data prior to clustering. Only SMCE reasonably separates the low-dimensional representation of face images according to the subjects. Note that in this experiment, the space of face images under varying illumination is not densely sampled and in addition the two manifolds are very close to each other. Table 2 shows the percentage of points in the dataset whose $K$ nearest neighbors contain points from the other manifold. As the table shows, there are several points whose closest neighbor comes from the other manifold. Beside the embedding of each method in Figure 4 (top row), we have shown the coefficient vector of a data point in $\mathcal{M}_1$ whose closest neighbor comes from $\mathcal{M}_2$. While nearest-neighbor-based methods pick the wrong neighbors with strong weights, SMCE successfully selects sparse neighbors from the correct manifold. The plots in the bottom of Figure 4 show the embedding obtained by SMCE for each cluster. As we move along the horizontal axis, the direction of the light source changes from left to right, while as we move along the vertical axis, the overall darkness of the images changes from light to dark. Also, the $\boldsymbol{msc}$ vectors suggest a 2-dimensionality of the face manifolds, correctly reflecting the number of degrees of freedom of the light source on the illumination rig, which is a sphere in $\mathbb{R}^3$.

Next, we consider the dimensionality reduction of the images in the Frey face dataset, which consists of 1965 face images captured under varying pose and expression. Each image is vectorized as a 560 element vector of pixel intensities. Figure 5 shows the two-dimensional embedding obtained by SMCE. Note that the low-dimensional representation captures well the left to right pose variations in the horizontal axis and the expression changes in the vertical axis.

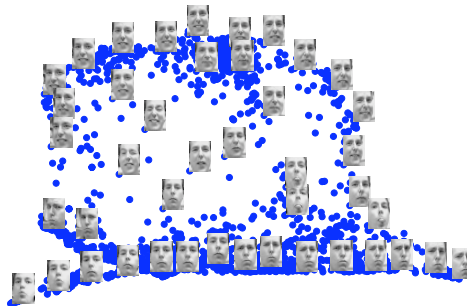

Figure 5: 2-D embedding of Frey face data using SMCE.

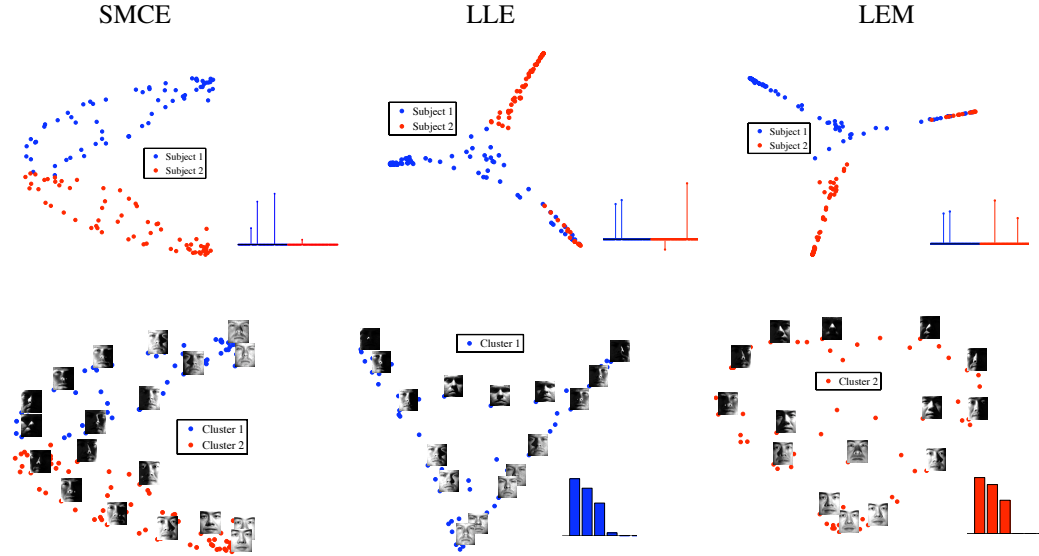

Figure 4: Clustering and embedding of two faces. Top: 2-D embedding obtained by SMCE, LLE and LEM. The weights associated to a data point from the first subject are shown beside the embedding. Bottom: SMCE embedding and *msc* vectors.

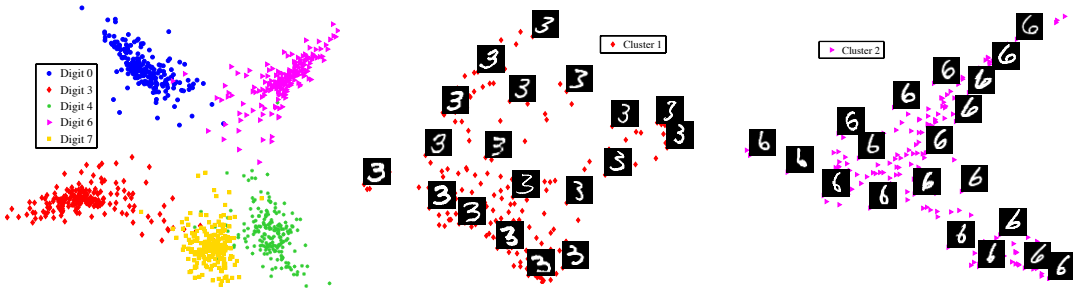

Figure 6: Clustering and embedding of five digits from the MNIST dataset. Left: 2-D embedding obtained by SMCE for five digits $\{0, 3, 4, 6, 7\}$. Middle: 2-D embedding of the data in the first cluster that corresponds to digit 3. Right: 2-D embedding of the data in the second cluster that corresponds to digit 6.

Finally, we consider the clustering and dimensionality reduction of the digits from the MNIST test database [23]. We use the images from five digits $\{0, 3, 4, 6, 7\}$ in the dataset where we randomly select 200 data points from each digit. The left plot in Figure 6 shows the joint embedding of the whole data using SMCE. One can see that the data are reasonably well separated according to their classes. The middle and the right plots in Figure 6, show the two-dimensional embedding obtained by SMCE for two data clusters, which correspond to the digits 3 and 6.

## 4 Discussion

We proposed a new algorithm based on sparse representation for simultaneous clustering and dimensionality reduction of data lying in multiple manifolds. We used the solution of a sparse optimization program to build a similarity graph from which we obtained clustering and low-dimensional embedding of the data. The sparse representation of each data point ideally encodes information that can be used for inferring the dimensionality of the underlying manifold around that point. Finding robust methods for estimating the intrinsic dimension of the manifolds from the sparse coefficients and investigating theoretical guarantees under which SMCE works is the subject of our future research.

## Acknowledgment

This work was partially supported by grants NSF CNS-0931805, NSF ECCS-0941463 and NSF OIA-0941362.

## Footnotes

[1]Note that a symmetric adjacency matrix can be obtained by taking $\boldsymbol{W} = \max(\boldsymbol{W}, \boldsymbol{W}^\top)$.

[2]One can also use the mean of the sorted coefficients in each cluster to compute the dimension of each manifold. However, we prefer to use the median for robustness reasons.

# References

[1] S. Roweis and L. Saul, "Nonlinear dimensionality reduction by locally linear embedding," *Science*, vol. 290, no. 5500, pp. 2323–2326, 2000.

[2] D. Donoho and C. Grimes, "Hessian eigenmaps: Locally linear embedding techniques for high-dimensional data," *National Academy of Sciences*, vol. 100, no. 10, pp. 5591–5596, 2003.

[3] M. Belkin and P. Niyogi, "Laplacian eigenmaps and spectral techniques for embedding and clustering," in *Neural Information Processing Systems*, 2002, pp. 585–591.

[4] J. B. Tenenbaum, V. de Silva, and J. C. Langford, "A global geometric framework for nonlinear dimensionality reduction," *Science*, vol. 290, no. 5500, pp. 2319–2323, 2000.

[5] K. Q. Weinberger and L. Saul, "Unsupervised learning of image manifolds by semidefinite programming," in *IEEE Conference on Computer Vision and Pattern Recognition*, 2004, pp. 988–955.

[6] B. Shaw and T. Jebara, "Minimum volume embedding," in *Artificial Intelligence and Statistics*, 2007.

[7] ——, "Structure preserving embedding," in *International Conference on Machine Learning*, 2009.

[8] R. Vidal, "Subspace clustering," *Signal Processing Magazine*, vol. 28, no. 2, pp. 52–68, 2011.

[9] D. Barbará and P. Chen, "Using the fractal dimension to cluster datasets," in *KDD '00: Proceedings of the sixth ACM SIGKDD international conference on Knowledge discovery and data mining*, 2000, pp. 260–264.

[10] P. Mordohai and G. G. Medioni, "Unsupervised dimensionality estimation and manifold learning in high-dimensional spaces by tensor voting." in *International Joint Conference on Artificial Intelligence*, 2005, pp. 798–803.

[11] A. Gionis, A. Hinneburg, S. Papadimitriou, and P. Tsaparas, "Dimension induced clustering," in *KDD '05: Proceeding of the eleventh ACM SIGKDD international conference on Knowledge discovery in data mining*, 2005, pp. 51–60.

[12] E. Levina and P. J. Bickel, "Maximum likelihood estimation of intrinsic dimension." in *NIPS*, 2004.

[13] G. Haro, G. Randall, and G. Sapiro, "Translated poisson mixture model for stratification learning," *International Journal of Computer Vision*, 2008.

[14] M. Polito and P. Perona, "Grouping and dimensionality reduction by locally linear embedding," in *Neural Information Processing Systems*, 2002.

[15] A. Goh and R. Vidal, "Segmenting motions of different types by unsupervised manifold clustering," in *IEEE Conference on Computer Vision and Pattern Recognition*, 2007.

[16] ——, "Clustering and dimensionality reduction on Riemannian manifolds," in *IEEE Conference on Computer Vision and Pattern Recognition*, 2008.

[17] D. Donoho and X. Huo, "Uncertainty principles and ideal atomic decomposition," *IEEE Trans. Information Theory*, vol. 47, no. 7, pp. 2845–2862, Nov. 2001.

[18] R. Tibshirani, "Regression shrinkage and selection via the lasso," *Journal of the Royal Statistical Society B*, vol. 58, no. 1, pp. 267–288, 1996.

[19] A. Ng, Y. Weiss, and M. Jordan, "On spectral clustering: analysis and an algorithm," in *Neural Information Processing Systems*, 2001, pp. 849–856.

[20] U. von Luxburg, "A tutorial on spectral clustering," *Statistics and Computing*, vol. 17, 2007.

[21] Z. Zhang and H. Zha, "Principal manifolds and nonlinear dimensionality reduction via tangent space alignment," *SIAM J. Sci. Comput.*, vol. 26, no. 1, pp. 313–338, 2005.

[22] K.-C. Lee, J. Ho, and D. Kriegman, "Acquiring linear subspaces for face recognition under variable lighting," *IEEE Transactions on Pattern Analysis and Machine Intelligence*, vol. 27, no. 5, pp. 684–698, 2005.

[23] Y. LeCun, L. Bottou, Y. Bengio, and P. Haffner, "Gradient-based learning applied to document recognition," in *Proceedings of the IEEE*, 1998, pp. 2278 – 2324.

